# Stable Fixed Points of Loopy Belief Propagation Are Minima of the Bethe Free Energy

**Tom Heskes**
SNN, University of Nijmegen
Geert Grooteplein 21, 6252 EZ, Nijmegen, The Netherlands

## Abstract

We extend recent work on the connection between loopy belief propagation and the Bethe free energy. Constrained minimization of the Bethe free energy can be turned into an unconstrained saddle-point problem. Both converging double-loop algorithms and standard loopy belief propagation can be interpreted as attempts to solve this saddle-point problem. Stability analysis then leads us to conclude that *stable* fixed points of loopy belief propagation must be (local) *minima* of the Bethe free energy. Perhaps surprisingly, the converse need not be the case: minima can be unstable fixed points. We illustrate this with an example and discuss implications.

## 1 Introduction

Pearl's belief propagation [1] is a popular algorithm for inference in Bayesian networks. It is exact in special cases, e.g., for tree-structured (singly-connected) networks with just Gaussian or just discrete nodes. But also on networks containing cycles, so-called loopy belief propagation often leads to good performance (approximate marginals close to exact marginals) [2]. The notion that fixed points of loopy belief propagation correspond to extrema of the so-called Bethe free energy [3] has been an important step in the theoretical understanding of this success. Empirically it has further been observed that loopy belief propagation, when it does, converges to a minimum. The main goal of this article is to understand why.

In Section 2 we will introduce loopy belief propagation in terms of a sum-product algorithm on factor graphs [4]. The corresponding Bethe free energy is derived in Section 3 from a variational point of view, indicating that we should be particularly interested in minima. In Section 4 we show that minimization of the Bethe free energy under the appropriate constraints is equivalent to an unconstrained saddle-point problem. The converging double-loop algorithm, described in Section 3, as well as the standard sum-product algorithm are in fact attempts to solve this saddle-point problem. More specifically, (a damped version of) the sum-product algorithm has the same local stability properties as a gradient descent-ascent procedure. Stability analysis of this gradient descent-ascent procedure then leads to the conclusion in the title. With an example we illustrate that the converse need not be the case. In Section 5 we discuss further implications and relations to other studies.

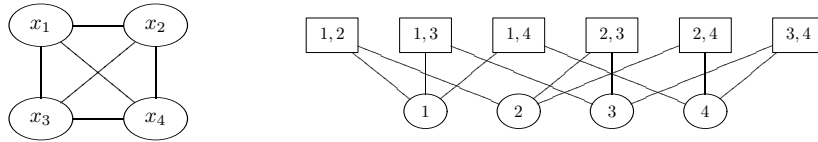

(a) Graphical model of

$P(x_1, \ldots, x_n) \propto$
$\exp \left[ \sum_{ij} w_{ij} x_i x_j + \sum_i \theta_i x_i \right].$

(b) Factor graph with potentials

$\Psi_{ij}(x_i, x_j) = \exp \left[ w_{ij} x_i x_j + \frac{1}{n-1} \theta_i x_i + \frac{1}{n-1} \theta_j x_j \right].$

Figure 1: A Boltzmann machine. (a) Graphical representation of the probability distribution. (b) Corresponding factor graph with a factor for each pair of nodes.

## 2   The sum-product algorithm on factor graphs

We start with a description of (loopy) belief propagation as the sum-product algorithm on factor graphs [4]. We assume that the probability distribution over (disjoint subsets of) variables $x_\beta$ factorizes over "factors" $\Psi_\alpha(X_\alpha)$:

$$P(x_1, \ldots, x_\beta, \ldots, x_N) = \frac{1}{Z} \prod_\alpha \Psi_\alpha(X_\alpha) \,, \qquad (1)$$

with $Z$ a proper normalization constant. We will use notation similar to [4]: upper-case $X_\alpha$ for the factors ("local function nodes") and lowercase $x_\beta$ for the variables. $\beta \subset \alpha$ means that $x_\beta$ is a neighbor of $X_\alpha$ in the factor graph, i.e., is included in the potential $\Psi_\alpha(X_\alpha)$. An example of the transformation of a Markov network into a factor graph is shown in Figure 1. In a similar manner one can transform Bayesian networks into factor graphs, where each factor contains the child and its parents [4].

On singly-connected structures, Pearl's belief propagation algorithm [1] can be applied to compute the exact marginals ("beliefs")

$$P(X_\alpha) = \sum_{X_{\setminus \alpha}} P(X) \quad \text{and} \quad P(x_\beta) = \sum_{X_{\setminus \beta}} P(X) \,.$$

If the structure contains cycles, one can still apply (loopy) belief propagation, in an attempt to obtain accurate approximations $P_\alpha(X_\alpha)$ and $P_\beta(x_\beta)$.

Pseudo-code for the sum-product algorithm is given in Algorithm 1. In the factor-graph representation we distinguish messages from factor $\alpha$ to variable $\beta$, $\mu_{\alpha \to \beta}(x_\beta)$, and vice versa, $\mu_{\beta \to \alpha}(x_\beta)$. The beliefs follow by multiplying the potential, a mere 1 for the variables and $\Psi_\alpha(X_\alpha)$ for the factors, with the incoming messages, see (1.3) and (1.2) in Algorithm 1. The update for an outgoing message is the variable belief, either calculated with the definition (1.2) or through the marginalization (1.6), divided by the incoming message, see (1.4) and (1.5).

We interpret the update of factor-variable message $\mu_{\alpha \to \beta}$ in line 8 of Algorithm 1 as the only actual update: beliefs and variable-factor messages directly follow from definitions in lines 11 to 15. For later reference we introduce the damped update

$$\log \mu_{\alpha \to \beta}^{\text{new}}(x_\beta) = \log \mu_{\alpha \to \beta}(x_\beta) + \epsilon \left[ \log \mu_{\alpha \to \beta}^{\text{full}}(x_\beta) - \log \mu_{\alpha \to \beta}(x_\beta) \right] \,, \qquad (2)$$

where $\mu^{\text{full}}$ refers to the result of the full update (1.5) and $\mu$ to the previous message. These and other seemingly arbitrary choices, among which the particular ordering

| | |
|---|---|
| 1: **repeat** | Initial messages: |
| 2: **for all** variables $\beta$ **do** | |
| 3: **for all** factors $\alpha \supset \beta$ **do** | $$\mu_{\alpha \to \beta}(x_\beta) = 1 \qquad (1.1)$$ |
| 4: **if** initial **then** | Beliefs: |
| 5: initialize message (1.1) | |
| 6: **else** | $$P_\beta(x_\beta) = \frac{1}{Z_\beta} \prod_{\alpha \supset \beta} \mu_{\alpha \to \beta}(x_\beta) \qquad (1.2)$$ |
| 7: marginalize (1.6) | |
| 8: update message (1.5) | $$P_\alpha(X_\alpha) = \frac{1}{Z_\alpha} \Psi_\alpha(X_\alpha) \prod_{\beta \subset \alpha} \mu_{\beta \to \alpha}(x_\beta) \ (1.3)$$ |
| 9: **end if** | |
| 10: **end for** | Messages: |
| 11: compute variable belief (1.2) | |
| 12: **for all** factors $\alpha \supset \beta$ **do** | $$\mu_{\beta \to \alpha}(x_\beta) = \frac{P_\beta(x_\beta)}{\mu_{\alpha \to \beta}(x_\beta)} \qquad (1.4)$$ |
| 13: compute message (1.4) | |
| 14: compute factor belief (1.3) | $$\mu_{\alpha \to \beta}(x_\beta) = \frac{P_\alpha(x_\beta)}{\mu_{\beta \to \alpha}(x_\beta)} \qquad (1.5)$$ |
| 15: **end for** | with |
| 16: **end for** | |
| 17: **until** convergence | $$P_\alpha(x_\beta) \equiv \sum_{X_{\alpha \setminus \beta}} P_\alpha(X_\alpha) \qquad (1.6)$$ |

Algorithm 1: The sum-product algorithm on factor graphs.

of updates, follow naturally from the analysis below. Besides, for the results on local stability we will consider the limit of small step sizes $\epsilon$, where any effects of the ordering disappear. Last but not least, the description in Algorithm 1 is mainly pedagogical and can be made more efficient in several ways.

## 3 The Bethe free energy

The exact distribution (1) can be written as the result of the variational problem

$$P(X) = \underset{\hat{P}}{\mathrm{argmin}} \sum_X \hat{P}(X) \log \left[ \frac{\hat{P}(X)}{\prod_\alpha \Psi_\alpha(X_\alpha)} \right], \qquad (3)$$

where here and in the following normalization and positivity constraints on probabilities are implicitly assumed. Next we confine our search to "tree-like" probability distributions of the form

$$\hat{P}(X) \propto \frac{\prod_\alpha P_\alpha(X_\alpha)}{\prod_\beta P_\beta(x_\beta)^{n_\beta - 1}} \text{ with } n_\beta \equiv \sum_{\alpha \supset \beta} 1, \qquad (4)$$

the number of neighboring factors of variable $\beta$. Here $P_\alpha(X_\alpha)$ and $P_\beta(x_\beta)$ are interpreted as (approximate) local marginals that should normalize to 1, but should also be consistent, i.e., obey

$$\forall_\beta \forall_{\alpha \supset \beta} \quad P_\alpha(x_\beta) = P_\beta(x_\beta), \qquad (5)$$

with $P_\alpha(x_\beta)$ as in (1.6). The denominator in (4) prevents double-counting. For singly-connected structures, it can be shown that the exact solution $P(X)$ is of this form, with proportionality constant equal to 1 and where $P_\alpha(X_\alpha) = P(X_\alpha)$ and $P_\beta(x_\beta) = P(x_\beta)$. For structures containing cycles, this need not be the case, but we can still assume it to be true approximately. Plugging (4) into the objective (3) and implementing the above assumptions, we obtain the Bethe free energy

$$F(P) = \sum_\alpha \sum_{X_\alpha} P_\alpha(X_\alpha) \log \left[ \frac{P_\alpha(X_\alpha)}{\Psi_\alpha(X_\alpha)} \right] - \sum_\beta (n_\beta - 1) \sum_{x_\beta} P_\beta(x_\beta) \log P_\beta(x_\beta). \quad (6)$$

<div style="border:1px solid">

| | Initial messages and beliefs: |
|---|---|
| 1: **for all** $\alpha$ and $\beta \subset \alpha$ **do** | $$\mu_{\beta \to \alpha}(x_\beta) = 1 \text{ and } P_\alpha(x_\beta) = 1 \quad (2.1)$$ |
| 2:   initialize (2.1) | Beliefs: |
| 3: **end for** | $$P_\beta(x_\beta) = \frac{1}{Z_\beta}\left[\prod_{\alpha \supset \beta}\mu_{\alpha \to \beta}(x_\beta)\right]^{\frac{1}{n_\beta}} \quad (2.2)$$ |
| 4: **repeat** | |
| 5:   **for all** factors $\alpha$ **do** | $$P_\alpha(X_\alpha) = \frac{1}{Z_\alpha}\hat{\Psi}_\alpha(X_\alpha)\prod_{\beta \subset \alpha}\mu_{\beta \to \alpha}(x_\beta) \,(2.3)$$ |
| 6:     update potential (2.4) | |
| 7:     update variable belief (2.3) | Potential update: |
| 8:   **end for** | $$\log\hat{\Psi}_\alpha(X_\alpha) = \log\Psi_\alpha(X_\alpha)$$ |
| 9:   inner loop with (2.2) and (2.3) | |
| 10: **until** convergence | $$+\sum_{\beta \subset \alpha}\frac{n_\beta - 1}{n_\beta}\log P_\alpha^{\text{old}}(x_\beta) \;(2.4)$$ |

</div>

Algorithm 2: Double-loop algorithm for minimizing the Bethe free energy. The inner loop is Algorithm 1 with redefinitions of the factor and variable beliefs.

Minus the Bethe free energy is an approximation, but not a bound of the loglikelihood $\log Z$. A key observation in [3] is that the fixed points of the sum-product algorithm, described in the previous section, correspond to extrema of the Bethe free energy under the constraints (5).

The above derivation suggests that we should be specifically interested in minima of the Bethe free energy, not "just" stationary points. The resulting constrained minimization problem is well-defined (the Bethe free energy is bounded from below), but not necessarily convex, mainly because of the negative $P_\beta \log P_\beta$-terms. The crucial trick, implicit or explicit in recently suggested procedures is to bound [5] or clamp [6] the possibly concave part (outer loop: recompute the bound) and solve the remaining convex problem (inner loop: maximization with respect to Lagrange multipliers; see below). Here we propose to use the linear bound

$$-\sum_{x_\beta}P_\beta(x_\beta)\log P_\beta(x_\beta) \le -\sum_{x_\beta}P_\beta(x_\beta)\log P_\beta^{\text{old}}(x_\beta)\,, \qquad (7)$$

with $P_\beta^{\text{old}}(x_\beta)$ from the result of the previous inner loop. The (convex) bound of the Bethe free energy then boils down to

$$F_{\text{bound}}(P) = \sum_\alpha \sum_{X_\alpha} P_\alpha(X_\alpha)\log\left[\frac{P_\alpha(X_\alpha)}{\hat{\Psi}_\alpha(X_\alpha)}\right] \ge F(P)\,,$$

if we define $\hat{\Psi}_\alpha$ as in (2.4). The outer loop corresponds to a reset of the bound, i.e., at the start of the inner loop we have $F_{\text{bound}}(P) = F(P)$. In the inner loop (see the next section for its derivation), we solve the remaining convex constrained minimization problem with the method of Lagrange multipliers. At the end of the inner loop, we then have $F(P^{\text{new}}) \le F_{\text{bound}}(P^{\text{new}}) \le F_{\text{bound}}(P) = F(P)$.

## 4 Saddle-point problem

In this section we will translate the (non-convex) minimization of the Bethe free energy under linear constraints into an equivalent (non-convex/concave) saddle-point

problem. We replace the bound (7) with an explicit minimization over auxiliary variables $\gamma$ (see also [7]; an alternative interpretation is a Legendre transform):

$$-\sum_{x_\beta} P_\beta(x_\beta) \log P_\beta(x_\beta) = \min_{\gamma_\beta} \left\{ -\sum_{x_\beta} \gamma_\beta(x_\beta) P_\beta(x_\beta) + \log \left[ \sum_{x_\beta} e^{\gamma_\beta(x_\beta)} \right] \right\}. \quad (8)$$

Substitution into (6) then yields a constrained minimization problem, where the minimization is w.r.t. $\{P_\alpha, P_\beta, \gamma_\beta\}$ under constraints (5). Using (any other convex combination will work as well, but this symmetric one is most convenient)

$$P_\beta(x_\beta) = \frac{1}{n_\beta} \sum_{\alpha \supset \beta} P_\alpha(x_\beta)$$

we can get rid of all dependencies on $P_\beta$, both in (8) and in the constraints (5), which simplifies the following analysis and derivations considerably. For fixed $\gamma_\beta$, the remaining minimization problem is convex in $P_\alpha$ with linear constraints and can thus be solved with the method of Lagrange multipliers. In terms of these multipliers $\lambda$ and the auxiliary variables $\gamma$, the solution for $P_\alpha$ reads

$$P_\alpha(X_\alpha) = \frac{1}{Z_\alpha(\lambda, \gamma)} \Psi_\alpha(X_\alpha) \exp \left[ \sum_{\beta \subset \alpha} \bar{\lambda}_{\alpha\beta}(x_\beta) + \frac{n_\beta - 1}{n_\beta} \gamma_\beta(x_\beta) \right], \quad (9)$$

with $Z_\alpha(\lambda, \gamma)$ the proper normalization and

$$\bar{\lambda}_{\alpha\beta}(x_\beta) \equiv \lambda_{\alpha\beta}(x_\beta) - \frac{1}{n_\beta} \sum_{\alpha' \supset \beta} \lambda_{\alpha'\beta}(x_\beta).$$

Substituting this back into the Lagrangian, we end up with an unconstrained saddle-point problem of the type $\min_\gamma \max_\lambda F(\lambda, \gamma)$ with

$$F(\lambda, \gamma) = \sum_\alpha \log Z_\alpha(\lambda, \gamma) - \sum_\beta (n_\beta - 1) \log \left[ \sum_{x_\beta} e^{\gamma_\beta(x_\beta)} \right].$$

From the fixed-point equations we derive the updates

$$\lambda_{\alpha\beta}^{\text{new}}(x_\beta) = \lambda_{\alpha\beta}(x_\beta) - \log P_\alpha(x_\beta) + \frac{1}{n_\beta} \sum_{\alpha' \supset \beta} \log P_{\alpha'}(x_\beta), \quad (10)$$

$$\gamma_\beta^{\text{new}}(x_\beta) = \log \left[ \frac{1}{n_\beta} \sum_{\alpha \supset \beta} P_\alpha(x_\beta) \right], \quad (11)$$

with $P_\alpha(x_\beta)$ the marginal computed from $P_\alpha(X_\alpha)$ as in (9).

**Proof.** Introduce a new set of auxiliary variables $\hat{Z}_\alpha$ by writing

$$-\log Z_\alpha = \max_{\hat{Z}_\alpha} \left\{ -\log \hat{Z}_\alpha + \left( 1 - \frac{1}{\hat{Z}_\alpha} \sum_{X_\alpha} P_\alpha(X_\alpha) Z_\alpha \right) \right\}.$$

Next consider maximizing $\lambda_{\alpha\beta}(x_\beta)$ for a particular variable $\beta$ and all $\alpha \supset \beta$, while keeping all others as well as all $\hat{Z}_\alpha$ fixed (by convention, we update $\hat{Z}_\alpha$ to $Z_\alpha$ after each update of $\lambda$'s). Taking derivatives, we find that the new $\bar{\lambda}^{\text{new}}$ should satisfy

$$\frac{e^{\bar{\lambda}_{\alpha\beta}^{\text{new}}(x_\beta)} P_\alpha(x_\beta)}{e^{\bar{\lambda}_{\alpha\beta}(x_\beta)}} = \frac{1}{n_\beta} \sum_{\alpha' \supset \beta} \frac{e^{\bar{\lambda}_{\alpha'\beta}^{\text{new}}(x_\beta)} P_{\alpha'}(x_\beta)}{e^{\bar{\lambda}_{\alpha'\beta}(x_\beta)}}.$$

Any update of the form $\lambda_{\alpha\beta}^{\text{new}}(x_\beta) = -\log P_\alpha(x_\beta) + \lambda_{\alpha\beta}(x_\beta) + \nu_\beta(x_\beta)$ will do, where choosing $\nu_\beta(x_\beta)$ such that $\lambda_{\alpha\beta}^{\text{new}} = \bar{\lambda}_{\alpha\beta}^{\text{new}}$ yields (10).

The updates (10) and (11) are properly aligned with the respective gradients and satisfy the saddle-point equations

$$F(\lambda^{\text{new}}, \gamma) \geq F(\lambda, \gamma) \geq F(\lambda, \gamma^{\text{new}}) . \tag{12}$$

This saddle-point problem is concave in $\lambda$, but not necessarily convex in $\gamma$. One way to guarantee convergence to a "correct" saddle point is then to solve the (up to irrelevant linear translations unique) maximization with respect to $\lambda$ in an inner loop, followed by an update of $\gamma$ in the outer loop. This is precisely the double-loop algorithm sketched in the previous section. We obtain the description given in Algorithm 2 if we substitute (up to irrelevant constants)

$$\gamma_\beta(x_\beta) = \log P_\beta^{\text{old}}(x_\beta), \ \bar{\lambda}_{\alpha\beta}(x_\beta) = \log \mu_{\beta\to\alpha}(x_\beta), \text{ and } \lambda_{\alpha\beta}(x_\beta) = -\log \mu_{\alpha\to\beta}(x_\beta) .$$

Note that in the inner loop of the double-loop algorithm the scheduling does matter. The ordering described in Algorithm 1 - run over variables $\beta$ and update all corresponding messages from and to neighboring factors before moving on to the next variable - satisfies (12) without damping.

An alternative approach is to apply (damped versions of) the updates (10) and (11) in parallel. This can be loosely interpreted as doing gradient descent-ascent. Gradient descent-ascent is a standard procedure for solving saddle-point problems and guaranteed to converge to the correct solution if the saddle-point problem is indeed convex/concave (see e.g. [8]). Similarly, it is easy to show that gradient descent-ascent applied to a non-convex/concave problem is locally stable at a particular saddle point $\{\lambda^*, \gamma^*\}$, if and only if the objective is locally convex/concave. The statement in the title now follows from two observations.

1. The damped version (2) of the sum-product algorithm has the same local stability properties as a gradient descent-ascent procedure derived from (10) and (11).

**Proof.** We replace (11) with

$$\gamma_\beta^{\text{new}}(x_\beta) = \frac{1}{n_\beta} \sum_{\alpha \supset \beta} \log P_\alpha(x_\beta) . \tag{13}$$

At a saddle point $P_\alpha(x_\beta) = P_\beta(x_\beta) \ \forall_{\alpha\supset\beta}$ and thus the difference between the logarithmic average (13) and the linear average (11) as well as its derivatives vanish. Consequently, (13) has the same local stability properties as (11). Now consider parallel application of a damped version of (10), with step size $\epsilon$, and (13), with step size $n_\beta\epsilon$. We obtain the damped version (2) of the standard sum-product algorithm, in combination with the other definitions in Algorithm 1, when we apply the definitions

$$\log \mu_{\beta\to\alpha}(x_\beta) = \bar{\lambda}_{\alpha\beta}(x_\beta) + \frac{n_\beta - 1}{n_\beta}\gamma_\beta(x_\beta) \text{ and } \log \mu_{\alpha\to\beta}(x_\beta) = \frac{1}{n_\beta}\gamma_\beta(x_\beta) - \lambda_{\alpha\beta}(x_\beta) .$$

2. Local stability of the gradient descent-ascent procedure at $\{\lambda^*, \gamma^*\}$ implies that the corresponding $P_\alpha$ is at a minimum of the Bethe free energy and that all constraints are satisfied. The converse need not be the case.

**Proof.** Local stability of the gradient descent-ascent procedure and thus the sum-product algorithm depends on the local curvature of $F(\lambda, \gamma)$, defined through the Hessian matrices

$$H_{\gamma\gamma} \equiv \left.\frac{\partial^2 F(\lambda, \gamma)}{\partial\gamma\partial\gamma^T}\right|_{\{\lambda^*, \gamma^*\}}$$

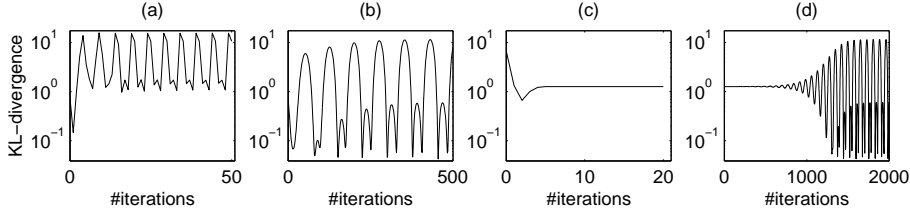

Figure 2: Loopy belief propagation on a Boltzmann machine with 4 nodes, weights (upper diagonal) $(3, 2, 2; 1, 3; -3)$, and thresholds $(0, 0, 1, 1)$. Plotted is the Kullback-Leibler divergence between the exact and the approximate single-node marginals. (a) No damping leads to somewhat erratic cyclic behavior. (b) Damping with step size 0.1 yields a smoother cycle, but no convergence. (c) The double-loop algorithm does converge to a stable solution. (d) This solution is unstable under standard loopy belief propagation (here again with step size 0.1).

and $H_{\lambda\lambda}$. Gradient descent-ascent is locally stable iff $H_{\gamma\gamma}$ is positive and $H_{\lambda\lambda}$ negative (semi-)definite. The latter is true by construction. The "total" curvature, defined through

$$H_{\gamma\gamma}^* \equiv \left.\frac{\partial^2 F^*(\gamma)}{\partial\gamma\partial\gamma^T}\right|_{\gamma^*} \quad \text{with } F^*(\gamma) \equiv \max_{\lambda} F(\lambda, \gamma),$$

can be shown to obey

$$H_{\gamma\gamma}^* = H_{\gamma\gamma} - H_{\gamma\lambda} H_{\lambda\lambda}^{-1} H_{\lambda\gamma}.$$

With $H_{\lambda\lambda}$ negative definite, we then conclude that if $H_{\gamma\gamma}$ is positive definite (gradient descent-ascent locally *stable*), then so is $H_{\gamma\gamma}^*$ (local *minimum*). The converse, however, need not be the case: $H_{\gamma\gamma}^*$ can be positive definite (minimum) where $H_{\gamma\gamma}$ has one or more negative eigenvalues (gradient descent-ascent unstable). An example of this phenomenom is $F(\lambda, \gamma) = -\lambda^2 - \gamma^2 + 4\lambda\gamma$.

Non-convergence of loopy belief propagation on a Boltzmann machine is shown in Figure 2. Typically, standard loopy belief propagation converges to a stable solution without damping. In rare cases, damping is required to obtain convergence and in very rare cases, even considerable damping does not help, as in Figure 2. The double-loop algorithm does converge and the solution obtained is indeed unstable under standard belief propagation, even with damping. The larger the weights, the more often these instabilities seem to occur. This is consistent with the empirical observation that the max-product algorithm ("belief revision") is typically less stable than the sum-product algorithm: max-product on a Boltzmann machine corresponds to (a properly scaled version of) the sum-product algorithm in the limit of infinite weights. The example in Figure 2 is about the smallest that we have found: we have observed these instabilities in many other (larger) instances of Markov networks, as well as directed Bayesian networks, yet not in structures with just a single loop. The latter seems consistent with the notion that not only for trees, but also for networks with a single loop, the Bethe free energy is still convex.

## 5    Discussion

The above gradient descent-ascent interpretation shows that loopy belief propagation is more than just fixed-point iteration: the updates tend to move in the right uphill-downhill directions, which might explain its success in practical applications. Still, loopy belief propagation can fail to converge, and apparently for two different

reasons. The first rather innocent one is a too large step size, similar to taking a too large "learning parameter" in gradient-descent learning. Straightforwardly damping the updates, as in (2), is then sufficient to converge to a stable fixed point. Note that this damping is in the logarithmic domain and thus slightly different from the damping linear in the messages as described in [2]. The damping proposed in [7] is restricted to the Lagrange multipliers $\lambda$ and may therefore not share the nice properties of the damping discussed here. Local stability in the limit of small step sizes is independent of the scheduling of messages, but in practice particular schedules can still favor others and, for example, be stable with larger step sizes or converge more rapidly. For example, in [9] the message updates follow the structure of a spanning tree, which empirically seems to help a lot.

The other more serious reason for non-convergence is inherent instability of the fixed point, even in the limit of infinitely small step sizes. In that case, loopy belief propagation just does not work and one can resort to a more tedious double-loop algorithm to guarantee convergence to a local minimum. The double-loop algorithm described here is similar to the CCCP algorithm of [5]. The latter implicitly uses a less strict bound, which makes it (slightly) less efficient and arguably a little more complicated. Whether double-loop algorithms are worth the effort is an open question: in several simulation studies a negative correlation between the quality of the approximation and the convergence of standard belief propagation has been found [6, 7, 10], but still without a convincing theoretical explanation.

## Acknowledgments

I would like to thank Wim Wiegerink and Onno Zoeter for many helpful suggestions and interesting discussions and the Dutch Technology Foundation STW for support.

## References

[1] J. Pearl. *Probabilistic Reasoning in Intelligent systems: Networks of Plausible Inference.* Morgan Kaufmann, San Francisco, CA, 1988.

[2] K. Murphy, Y. Weiss, and M. Jordan. Loopy belief propagation for approximate inference: An empirical study. In *UAI'99*, pages 467–475, 1999.

[3] J. Yedidia, W. Freeman, and Y. Weiss. Generalized belief propagation. In *NIPS 13*, pages 689–695, 2001.

[4] F. Kschischang, B. Frey, and H. Loeliger. Factor graphs and the sum-product algorithm. *IEEE Transactions on Information Theory*, 47(2):498–519, 2001.

[5] A. Yuille. CCCP algorithms to minimize the Bethe and Kikuchi free energies: Convergent alternatives to belief propagation. *Neural Computation*, 14:1691–1722, 2002.

[6] Y. Teh and M. Welling. The unified propagation and scaling algorithm. In *NIPS 14*, 2002.

[7] T. Minka. The EP energy function and minimization schemes. Technical report, MIT Media Lab, 2001.

[8] S. Seung, T. Richardson, J. Lagarias, and J. Hopfield. Minimax and Hamiltonian dynamics of excitatory-inhibitory networks. In *NIPS 10*, 1998.

[9] M. Wainwright, T. Jaakola, and A. Willsky. Tree-based reparameterization for approximate estimation on loopy graphs. In *NIPS 14*, 2002.

[10] T. Heskes and O. Zoeter. Expectation propagation for approximate inference in dynamic Bayesian networks. In *UAI-2002*, pages 216–223, 2002.
